# A Formal Model of the Insect Olfactory Macroglomerulus: Simulations and Analytical Results.

**Christiane Linster**
**David Marsan**
ESPCI, Laboratoire d'Electronique
10, Rue Vauquelin
75005 Paris, France

**Claudine Masson**
Laboratoire de Neurobiologie Comparée des
Invertébrées
INRA/CNRS (URA 1190)
91140 Bures sur Yvette, France

**Michel Kerszberg**
Institut Pasteur
CNRS (URA 1284)
Neurobiologie Moléculaire
25, Rue du Dr. Roux
75015 Paris, France

**Gérard Dreyfus**
**Léon Personnaz**
ESPCI, Laboratoire d'Electronique
10, Rue Vauquelin
75005 Paris, France

## Abstract

It is known from biological data that the response patterns of interneurons in the olfactory macroglomerulus (MGC) of insects are of central importance for the coding of the olfactory signal. We propose an analytically tractable model of the MGC which allows us to relate the distribution of response patterns to the architecture of the network.

## 1. Introduction

The processing of pheromone odors in the antennal lobe of several insect species relies on a number of response patterns of the antennal lobe neurons in reaction to stimulation with pheromone components and blends. Antennal lobe interneurons receive input from different receptor types, and relay this input to antennal lobe projection neurons via excitatory as well as inhibitory synapses. The diversity of the responses of the interneurons and projection neurons as well the long response latencies of these neurons to pheromone stimulation or electrical stimulation of the antenna, suggest a polysynaptic pathway

between the receptor neurons and these projection neurons (for a review see (Kaissling, 1990; Masson and Mustaparta, 1990)).

### I. PHEROMONE GENERALISTS

#### A. Cannot Discriminate Single Odors     AND     Cannot Code Temporal Changes

##### 1. Excited Type

| Stimulus | : | Response |
|----------|---|----------|
| BAL  | | + |
| C15  | | + |
| Blend | | + |

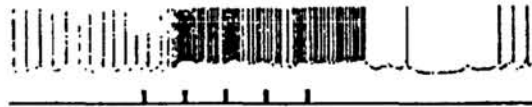

##### 2. Inhibited Type

| Stimulus | : | Response |
|----------|---|----------|
| BAL  | | - |
| C15  | | - |
| Blend | | - |

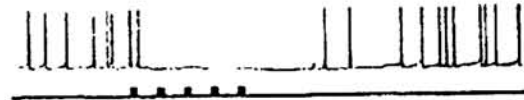

### II. PHEROMONE SPECIALISTS

#### A. Can Discriminate Single Odors     BUT     Cannot Code Temporal Changes

| Stimulus | : | Response | |
|----------|---|----------|---|
| | | (1) OR | (2) |
| BAL  | | + | 0 |
| C15  | | 0 | + |
| Blend | | + | + |

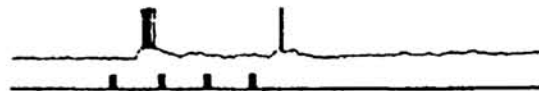

#### B. Can Discriminate Single Odors     AND     Can Code Temporal Changes

| Stimulus | : | Response | |
|----------|---|----------|---|
| | | (1) OR | (2) |
| BAL  | | + | - |
| C15  | | - | + |
| Blend | | -/+/- | -/+/- |

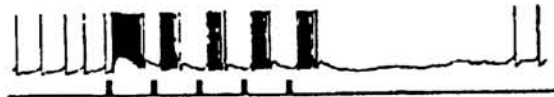

Figure 1: With courtesy of John Hildebrand, by permission from Oxford University Press, from: Christensen, Mustaparta and Hildebrand: Discrimination of sex pheromone blends in the olfactory system of the moth, Chemical Senses, Vol 14, no 3, pp 463-477, 1989.

In the MGC of *Manduca sexta*, antennal lobe interneurons respond in various ways to antennal stimulation with single pheromone components or the blend: pheromone generalists respond by either excitation or inhibition to both components and the blend: they cannot discriminate the components; pheromone specialists respond (i) to one component but not to the other by either excitation or inhibition, (ii) with different response patterns to the presence of the single components or the blend, namely with excitation to one component, with inhibition to the other component and with a mixed response to the blend. These neurons can also follow pulsed stimulation up to a cut-off frequency (Figure 1).

A model of the MGC (Linster et al, 1993), based on biological data (anatomical and physiological) has demonstrated that the full diversity of response patterns can be reproduced with a random architecture using very simple ingredients such as spiking neurons governed by a first order differential equation, and synapses modeled as simple delay lines. In a model with uniform distributions of afferent, inhibitory and excitatory synapses, the distribution of the response patterns depends on the following network parameters: the percentage of afferent, inhibitory and excitatory synapses, the ratio of the average excitation of any interneuron to its spiking threshold, and the amount of feedback in the network.

In the present paper, we show that the behavior of such a model can be described by a statistical approach, allowing us to search through parameter space and to make predictions about the biological system without exhaustive simulations. We compare the results obtained with simulation of the network model to the results obtained analytically by the statistical approach, and we show that the approximations made for the statistical descriptions are valid.

## 2. Simulations and comparison to biological data

In (Linster et al, 1993), we have used a simple neuron model: all neurons are spiking neurons, governed by a first order differential equation, with a membrane time constant and a probabilistic threshold $\Theta$. The time constant represents the decay time of the membrane potential of the neuron. The output of each neuron consists of an all-or-none action potential with unit amplitude that is generated when the membrane potential of the cell crosses a threshold, whose cumulative distribution function is a continuous and bounded probabilistic function of the membrane potential. All sources of delay and signal transformation from the presynaptic neuron to its postsynaptic site are modeled by a synaptic time delay. These delays are chosen in a random distribution (gaussian), with a longer mean value for inhibitory synapses than for excitatory synapses. We model two main populations of olfactory neurons: *receptor neurons* which are sensitive to the main pheromone component (called A) or to the minor pheromone component (called B) project uniformly onto the network of *interneurons*; two types of interneurons (excitatory and inhibitory) exist: each interneuron is allowed to make one synapse with any other interneuron.

The model exhibits several behaviors that agree with biological data, and it allows us to state several predictive hypotheses about the processing of the pheromone blend. We observe two broad classes of interneurons: selective (to one odor component) and non-selective neurons (in comparison to Figure 1). Selective neurons and non-selective neurons exhibit a variety of response patterns, which fall into three classes: inhibitory, excitatory and mixed (Figure 2). Such a classification has indeed been proposed for olfactory antennal

lobe neurons (local interneurons and projection neurons) in the specialist olfactory system in *Manduca* (Christensen and Hildebrand, 1987) and for the *cockroach* (Burrows et al, 1982; Boeckh and Ernst, 1987).

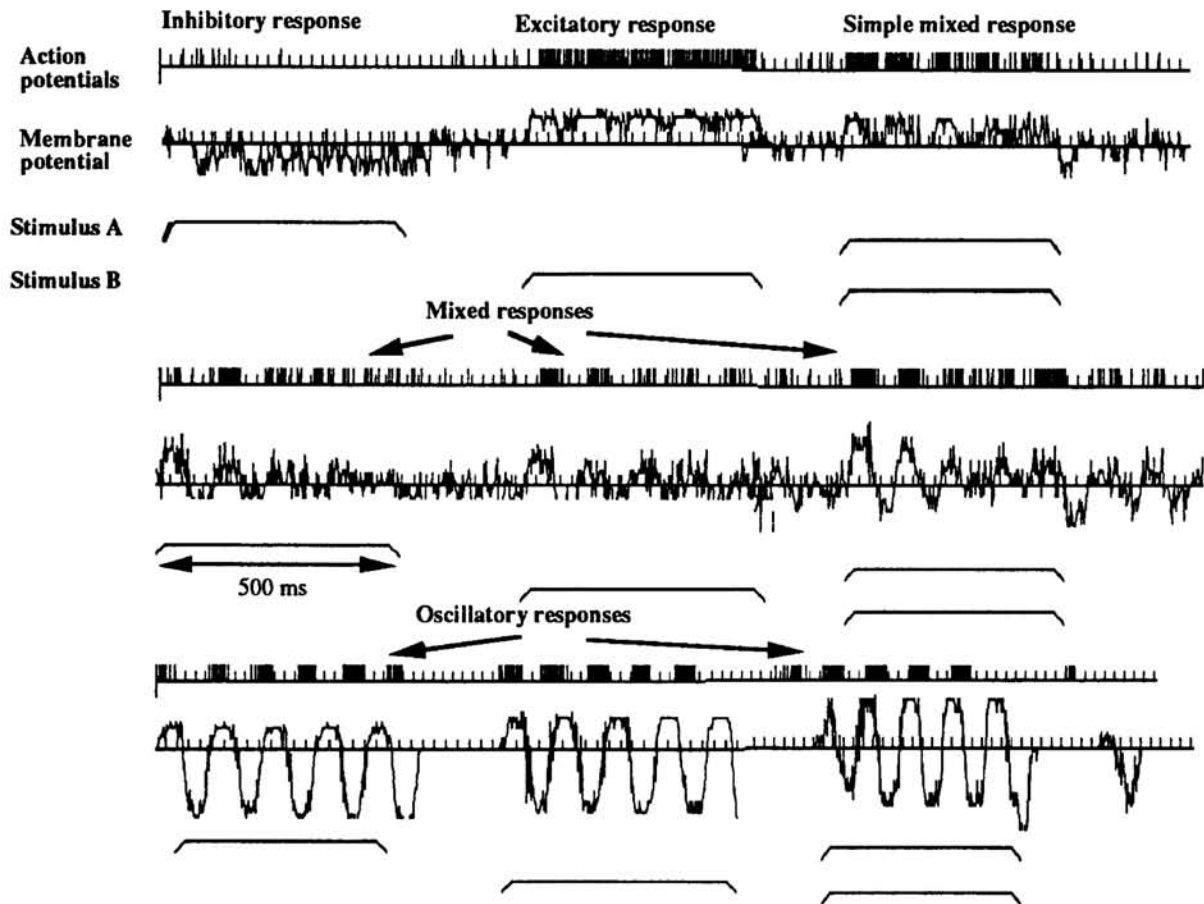

Figure 2: Response patterns of interneurons in the model presented, in response to stimulation with single components A and B, and with a blend with equal component concentrations. Receptor neurons fire at maximum frequency during the stimulations. The interneuron in the upper row is inhibited by stimulus A, excited by stimulus B, and has a mixed response (excitation followed by inhibition) to the blend: in reference to Figure 1, this is a pheromone specialist receiving mixed input from both types of receptor neurons. These types of simple and mixed responses can be observed in the model at low connectivity, where the average excitation received by an interneuron is low compared to its spiking threshold. The neuron in the middle row responds with similar mixed responses to stimuli A, B and A+B. The neuron in the lower row responds to all stimuli with the same oscillatory response, here the average excitation received by an interneuron approaches or exceeds the spiking threshold of the neurons. Network parameters: 15 receptor neurons; 35 interneurons; 40% excitatory interneurons; 60% inhibitory interneurons; afferent connectivity 10%; membrane time constant 25 ms; mean inhibitory synaptic delays 100 ms; mean excitatory synaptic delays 25 ms, spiking threshold 4.0, synaptic weights +1 and -1.

In our model, as well as in biological systems (Christensen and Hildebrand 1988, Christensen et al., 1989) we observe a number of local interneurons that cannot follow pulsed stimulation beyond a neuron-specific cut-off frequency. This frequency depends on the neuron response pattern and on the duration of the interstimulus interval.

Therefore, the type of response pattern is of central importance for the coding of the olfactory signal. Thus, in order to be able to relate the coding capabilities of a (model or biological) network to its architecture, we have investigated the distribution of response patterns both analytically and by simulations.

## 3. Analytical approach

In order to investigate these questions in a more rigorous way, some of us (C.L., D.M., G.D., L.P.) have designed a simplified, analytically tractable model.

We define two layers of interneurons: those which receive direct afferent input from the receptor neurons (layer 1), and those which receive only input from other interneurons (layer 2). In order to predict the response pattern of any interneuron as a function of the network parameters, we make the following assumptions: (i) statistically, all interneurons within a given layer receive the same synaptic input, (ii) the effect of feedback loops from layer 2 can be neglected, (iii) the response patterns have the same distribution for stimulations either by the blend or by pure components. Assumption (i) is correct because of the uniform distribution of synapses in the network of interneurons. Assumption (ii) is valid at low connectivity: if the average amount of excitation received by an interneuron is low as compared to its spiking threshold, its firing probability is low; therefore, the effect of the excitation from the receptors is vanishingly small beyond two interneurons: we thus neglect the effect of signals sent from layer 2. Thus, feedback is present within layer 1, and layer 2 receives only feedforward connections. Assumption (iii) is plausible if we suppose that the natural pheromone blend is more relevant for the system than the single components of the blend. We further assume in the analytical approach (as in the simulations) that the synaptic delays are longer on the average for inhibitory synapses than for excitatory synapses .

An interneuron can thus respond with four types of patterns: *non-response*, which means that it does not have a presynaptic neuron (this response pattern can only occur in layer 2, at low connectivity); *excitation*, meaning that an interneuron receives only afferent input from receptor neurons or from excitatory interneurons; *inhibition*, meaning that an interneuron receives only input from inhibitory interneurons (this can occur in layer 2 only); and *mixed responses*, covering all other combinations of presynaptic input.

We consider a network of $N + N_r$ neurons, $N$ (number of interneurons) and $N_r$ (number of receptor neurons) being random variables, $N + N_r$ being fixed. We define the probability $n_i$ that a neuron is an inhibitory interneuron, and the probability $n_e$ that it is an excitatory interneuron. Any interneuron has a probability $c$ to make one synapse (with synaptic weight +1 or -1) with any other interneuron and a probability $(1 - c)$ not to make a synapse with this interneuron; $c_r$ is the afferent connectivity: any receptor neuron has a probability $c_r$ to connect once to any interneuron, and a probability $(1 - c_r)$ not to connect to this interneuron. Then $n_a = 1 - (1 - c_r)^{N_r}$ is the probability that an interneuron belongs to layer 1, and the number of interneurons in layer 1 obeys a binomial distribution with expectation value $N n_a$ and variance $N n_a (1 - n_a)$. In the following, the fixed number of interneurons in layer 1 will be taken equal to its expectation value. Similarly, the number of interneurons in layer 2 is taken to be $N (1 - n_a)$.

Because of the assumptions made above, in both layers, we take into account for each interneuron the $N\,n_a\,c$ synapses from presynaptic neurons of layer1. In layer 1, these neurons respond with excitatory or mixed responses. $P_e^1 = n_e n_a\,N\,c$ is the probability that an interneuron in layer 1 responds with an excitation, and $P_m^1 = 1 - n_e n_a N\,c$ is the probability that an interneuron in layer 1 receives mixed synaptic input.

In layer 2, we have to consider two cases: (i) at low connectivity, if $N\,c\,n_a < 1$, $P_0^2 = 1 - N\,c\,n_a$ is the probability that an interneuron of layer 2 does not receive a synapse, thus does not respond to stimulation, $P_e^2 = N\,c\,n_a n_e$ is the probability that a neuron in layer 2 responds with excitation, $P_i^2 = N\,c\,n_a n_i$ is the probability that an interneuron responds with inhibition; (ii) at higher connectivity, $N\,c\,n_a > 1$, $P_0^2 = 0$, $P_e^2 = n_e\,n_a N\,c$ and $P_i^2 = n_i\,n_a N\,c$. In both cases (i) and (ii), the probability that an interneuron in layer 2 has a mixed response pattern is $P_m^2 = 1 - P_0^2 - P_e^2 - P_i^2$.

Thus, an interneuron in the model responds with excitation with probability $P_e = n_a\,P_e^1 + (1 - n_a)\,P_e^2$, with inhibition with probability $P_i = n_a\,P_i^1 + (1 - n_a)\,P_i^2$ and has a mixed response with probability $P_m = n_a\,P_m^1 + (1 - n_a)\,P_m^2$.

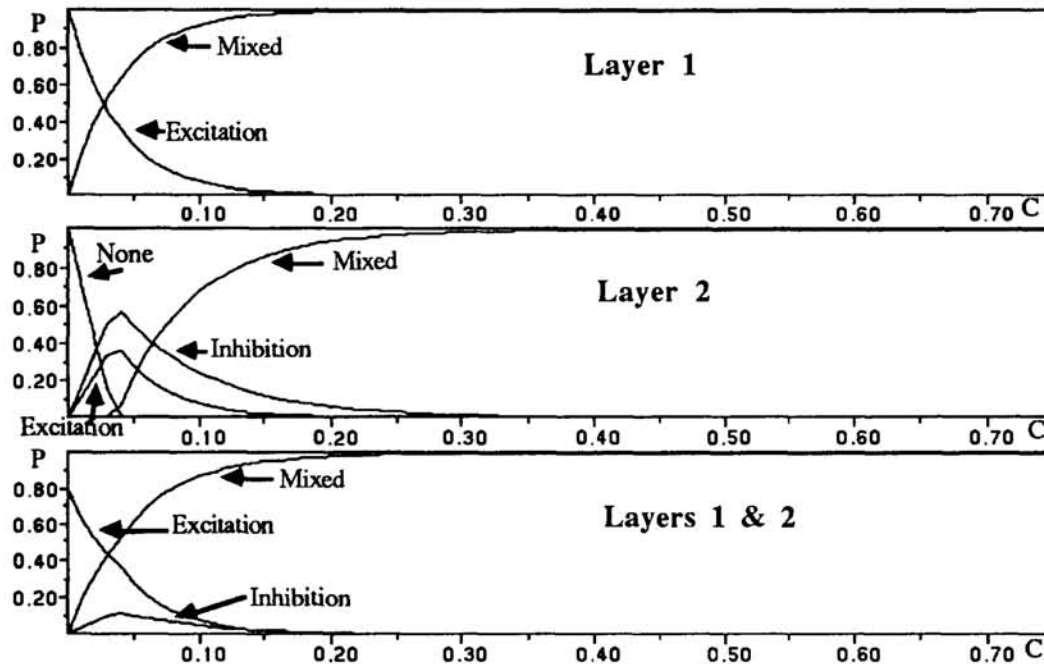

Figure 4: Analytically derived distribution of the response patterns in a typical network (35 interneurons, 15 receptor neurons, 40% excitation, 60% inhibition, spiking threshold 4.0); the curves show the percentage of interneurons in the model which respond with a given pattern, as a function of the connectivity c. In this case, the average excitation an interneuron receives from other interneurons is 3.15 at c=0.3.

Figure 4 shows the distribution of the response patterns computed analytically for a typical set of parameters. In order to perform comparisons between computed pattern distributions and pattern distributions obtained from simulations with the model, we designed an automatic classifier for the response patterns, based on the perceptron learning rule and the pocket algorithm (Gallant 1986). The classifier is trained to classify the responses of

individual interneurons, based on their membrane potential, into 5 typical response classes: non-response, pure excitation, pure inhibition, simple mixed response and oscillatory responses. Figure 5 shows the simulation results for the same set of parameters as for Figure 4. The agreement between the two curves shows that the approximations which we have made in order to describe the analytical model are valid.

Figure 6 shows how the mixed responses in the simulations divide into simple mixed and oscillatory responses. When the validity limit of the approximations made in the analytical approach is reached, all neurons fire at maximum frequency and the network oscillates. Therefore, the analytical model describes satisfactorily the whole range of connectivity in which the pattern distribution does not reduce to oscillations. The oscillation frequency is determined by the mean synaptic delays and by the membrane time constants; more detailed results on the oscillatory behavior will be published in a future paper.

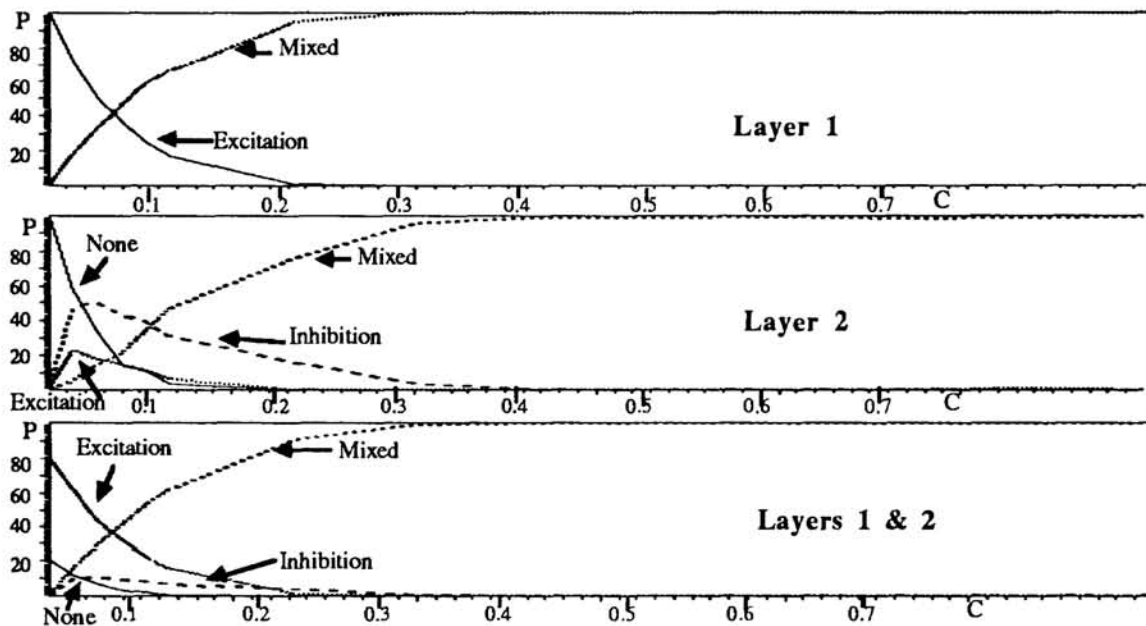

Figure 5: Distribution of the response patterns obtained from simulations of the model with the set of parameters described above. The curves show the percentages of interneurons that respond with a given pattern, as a function of connectivity c. For each value of c, 100 simulation runs with three different stimulation inputs have been averaged.

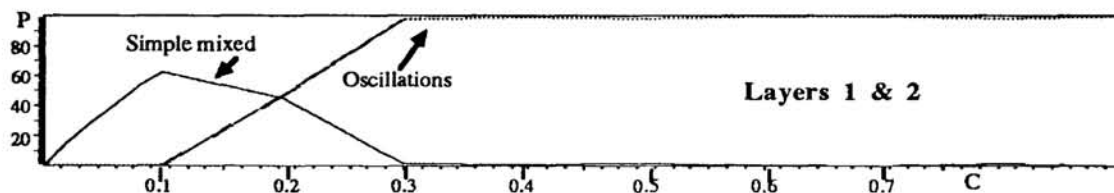

Figure 6: Distribution of simple mixed and oscillatory responses in the simulation model. With the set of parameters chosen, condition $n_e c \approx \Theta$ is satisfied for $c \approx 0.3$.

## 4. Conclusion

In the olfactory system of insects and mammals, a number of response patterns are observed, which are of central importance for the coding of the olfactory signal. In the present paper, we show that, under some constraints, an analytical model can predict the existence and the distribution of these response patterns. We further show that the transition between non-oscillatory and oscillatory regimes is governed by a single parameter ($n_e c / \Theta$). It is thus possible, to explore the parameter space without exhaustive simulations, and to relate the coding capabilities of a model or biological network to its architecture.

**Acknowledgements**
This work was supported in part by a grant from Ministère de la Recherche et de la Technologie (Sciences de la Cognition). C. Linster has been supported by a research grant (BFR91/051) from the Ministère des Affaires Culturelles, Grand-Duché de Luxembourg.

**References**
Boeckh, J. and Ernst, K.D. (1987). Contribution of single unit analysis in insects to an understanding of olfactory function. *J. Comp. Physiolo.* A161:549-565.
Burrows, M., Boeckh, J., Esslen, J. (1982). Physiological and Morphological Properties of Interneurons in the Deutocerebrum of Male Cockroaches which respond to Female Pheromone. *J. Comp. Physiolo.* 145:447-457.
Christensen, T.A., Hildebrand, J.G. (1987). Functions, Organization, and Physiology of the Olfactory Pathways in the Lepidoteran Brain. *In Arthropod Brain: its Evolution, Development, Structure and Functions,* A.P. Gupta, (ed), John Wiley & Sons.
Christensen, T.A., Hildebrand, J.G. (1988). Frequency coding by central olfactory neurons in the spinx moth Manduca sexta. *Chemical Senses* 13 (1):123-130.
Christensen, T.A., Mustaparta, H., Hildebrand, J.G. (1989). Discrimination of sex pheromone blends in the olfactory system of the moth. *Chemical Senses* 14 (3):463-477.
Kaissling, K-E., Kramer, E. (1990). Sensory basis of pheromone-mediated orientation in moths. *Verh. Dtsch. Zoolo. Ges.* 83:109-131.
Linster, C., Masson, C., Kerszberg, M., Personnaz, L., Dreyfus, G. (1993). Computational Diversity in a Formal Model of the Insect Olfactory Macroglomerulus. *Neural Computation* 5:239-252.
Masson, C., Mustaparta, H. (1990). Chemical Information Processing in the Olfactory System of Insects. *Physiol. Reviews* 70 (1):199-245.